# 'Ensemble' Boltzmann Units
# have Collective Computational Properties
# like those of Hopfield and Tank Neurons

Mark Derthick and Joe Tebelskis
Department of Computer Science
Carnegie-Mellon University

## 1   Introduction

There are three existing connectionist models in which network states are assigned a computational energy. These models—Hopfield nets, Hopfield and Tank nets, and Boltzmann Machines—search for states with minimal energy. Every link in the network can be thought of as imposing a constraint on acceptable states, and each violation adds to the total energy. This is convenient for the designer because constraint satisfaction problems can be mapped easily onto a network. Multiple constraints can be superposed, and those states satisfying the most constraints will have the lowest energy.

Of course there is no free lunch. Constraint satisfaction problems are generally combinatorial and remain so even with a parallel implementation. Indeed, Merrick Furst (personal communication) has shown that an NP-complete problem, graph coloring, can be reduced to deciding whether a connectionist network has a state with an energy of zero (or below). Therefore designing a practical network for solving a problem requires more than simply putting the energy minima in the right places. The topography of the energy space affects the ease with which a network can find good solutions. If the problem has highly interacting constraints, there will be many local minima separated by energy barriers. There are two principal approaches to searching these spaces: monotonic gradient descent, introduced by Hopfield [1] and refined by Hopfield and Tank [2]; and stochastic gradient descent, used by the Boltzmann Machine [3]. While the monotonic methods are not guaranteed to find the optimal solution, they generally find good solutions much faster than the Boltzmann Machine. This paper adds a refinement to the Boltzmann Machine search algorithm analogous to the Hopfield and Tank technique, allowing the user to trade off the speed of search for the quality of the solution.

# 2 Hopfield nets

A Hopfield net [1] consists of binary-valued units connected by symmetric weighted links. The global energy of the network is defined to be

$$E = -\frac{1}{2}\sum_i\sum_{j \neq i} w_{ij}s_is_j - \sum_i I_is_i$$

where $s_i$ is the state of unit $i$, and $w_{ij}$ is the weight on the link between units $i$ and $j$.

The search algorithm is: randomly select a unit and probe it until quiescence. During a probe, a unit decides whether to be on or off, determined by the states of its neighbors. When a unit is probed, there are two possible resulting global states. The difference in energy between these states is called the unit's *energy gap*:

$$\Delta_k \equiv E_{s_k=0} - E_{s_k=1} = \sum_i w_{ik}s_i + I_k$$

The decision rule is

$$s_i = \begin{cases} 0 \text{ if } \Delta_i < 0 \\ 1 \text{ otherwise} \end{cases}$$

This rule chooses the state with lower energy. With time, the global energy of the network monotonically decreases. Since there are only a finite number of states, the network must eventually reach quiescence.

# 3 Boltzmann Machines

A Boltzmann Machine [3] also has binary units and weighted links, and the same energy function is used. Boltzmann Machines also have a learning rule for updating weights, but it is not used in this paper. Here the important difference is in the decision rule, which is stochastic. As in probing a Hopfield unit, the energy gap is determined. It is used to determine a probability of adopting the on state:

$$P(s_i = 1) = \frac{1}{1 + e^{-\Delta_i/T}}$$

where $T$ is the computational temperature. With this rule, energy does not decrease monotonically. The network is more likely to adopt low energy states, but it sometimes goes uphill. The idea is that it can search a number of minima, but spends more time in deeper ones. At low temperatures, the ratio of time spent in the deepest minima is so large that the chances of not being in the global minimum are negligible. It has been proven [4] that after searching long enough, the probabilities of the states are given by the Boltzmann distribution, which is strictly a function of energy and temperature, and is independent of topography:

$$\frac{P_\alpha}{P_\beta} = e^{-(E_\alpha - E_\beta)/T} \tag{1}$$

The approach to equilibrium, where equation 1 holds, is speeded by initially searching at a high temperature and gradually decreasing it. Unfortunately, reaching equilibrium stills takes exponential time. While the Hopfield net settles quickly and is not guaranteed to find the best solution, a Boltzmann Machine can theoretically be run long enough to guarantee that the global optimum is found. Most of the time the uphill moves which allow the network to escape local minima are a waste of time, however. It is a direct consequence of the guaranteed ability to find the best solution that makes finding even approximate solutions slow.

## 4   Hopfield and Tank networks

In Hopfield and Tank nets [2], the units take on continuous values between zero and one, so the search takes place in the interior of a hypercube rather than only on its vertices. The search algorithm is deterministic gradient descent. By beginning near the center of the space and searching in the direction of steepest descent, it seems likely that the deepest minimum will be found. There is still no guarantee, but good results have been reported for many problems.

The modified energy equation is

$$E = -\frac{1}{2}\sum_i \sum_j w_{ij}s_i s_j + \sum_i \frac{1}{R_i}\int_0^{s_i} g^{-1}(s)ds - \sum_i I_i s_i \tag{2}$$

$R_i$ is the input resistance to unit $i$, and $g(u)$ is the sigmoidal unit transfer function $\frac{1}{1+e^{2\lambda u}}$. The second term is zero for extreme values of $s_i$, and is minimized at $s_i = \frac{1}{2}$.

The Hopfield and Tank model is continuous in time as well as value. Instead of proceeding by discrete probes, the system is described by simultaneous differential equations, one for each unit. Hopfield and Tank show that the following equation of motion results in a monotonic decrease in the value of the energy function:

$$\frac{du_i}{dt} = -u_i/\tau + \sum_j w_{ij}s_j + I_i$$

where $\tau = RC$, $C$ is a constant determining the speed of convergence, $u_i = g^{-1}(s_i)$, and the gain, $\lambda$, is analgous to (the inverse of) temperature in a Boltzmann Machine. $\lambda$ determines how important it is to satisfy the constraints imposed by the links to other units. When $\lambda$ is low, these constraints are largely ignored and the second term dominates, tending to keep the system near the center of the search space, where there is a single global minimum. At high gains, the minima lie at the corners of the search space, in the same locations as for the Hopfield model and the Boltzmann model. If the system is run at high gain, but the initial state is near the center of the space, the search gradually moves out towards the corners, on the way encountering "continental divides" between watersheds leading to all the various local minima. The initial steepness of the watersheds serves as a heuristic for choosing which minima is

likely to be lower. This search heuristic emerges automatically from the architecture, making network design simple. For many problems this single automatic heuristic results in a system comparable to the best knowledge intensive algorithms in which many domain specific heuristics are laboriously hand programmed.

For many problems, Hopfield and Tank nets seem quite sufficient [5, 6]. However for one network we have been using [7] the Hopfield and Tank model invariably settles into poor local minima. The solution has been to use a new model combining the advantages of Boltzmann Machines and Hopfield and Tank networks.

# 5   'Ensemble' Boltzmann Machines

It seems the Hopfield and Tank model gets its advantage by measuring the actual gradient, giving the steepest direction to move. This is much more informative than picking a random direction and deciding which of the two corners of the space to try, as models using binary units must do. Peter Brown (personal communication) has investigated continuous Boltzmann Machines, in which units stochastically adopt a state between zero and one. The scheme presented here has a similar effect, but the units actually take on discrete states between zero and one. Each ensemble unit can be thought of as an ensemble of identically connected conventional Boltzmann units. To probe the ensemble unit, each of its constituents is probed, and the state of the ensemble unit is the average of its constituents' states. Because this average is over a number of identical independent binary random variables, the ensemble unit's state is binomially distributed.

Figure 1 shows an ensemble unit with three constituents. At infinite temperature, all unit states tend toward $\frac{1}{2}$, and at zero temperature the states go to zero or one unless the energy gap is exactly zero. This is similar to the behavior of a Hopfield and Tank network at low and high gain, respectively. In Ensemble Boltzmann Machines (EBMs) the tendency towards $\frac{1}{2}$ in the absence of constraints from other units results from the shape of the binomial distribution. In contrast, the second term in the energy equation is responsible for this effect in the Hopfield and Tank model.

Although an EBM proceeds in discrete time using probes, over a large number of probes the search tends to proceed in the direction of the gradient. Every time a unit is probed, a move is made along one axis whose length depends on the magnitude of the gradient in that direction. Because probing still contains a degree of stochasticity, EBMs can escape from local minima, and if run long enough are guaranteed to find the global minimum. By varying $n$, the number of components of each ensemble unit, the system can exhibit any intermediate behavior in the tradeoff between the speed of convergence of Hopfield and Tank networks, and the ability to escape local minima of Boltzmann Machines.

Clearly when $n = 1$ the performance is identical to a conventional Boltzmann Machine, because each unit consists of a single Boltzmann unit. As $n \to \infty$ the

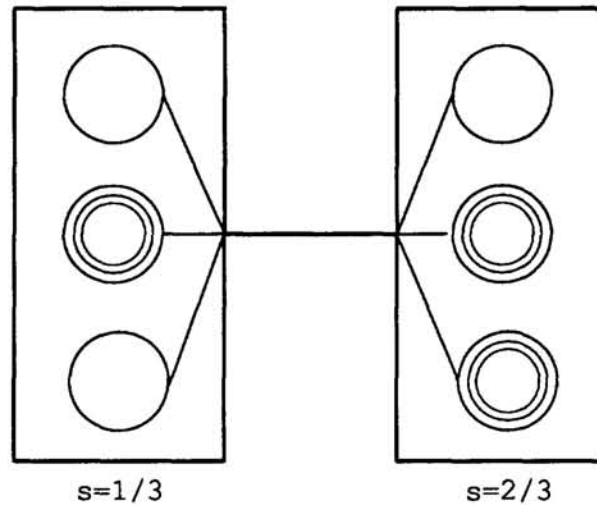

s=1/3  s=2/3

**Figure 1**: The heavy lines depict an 'Ensemble' Boltzmann Machine with two units. With an ensemble size of three, this network behaves like a conventional Boltzmann Machine consisting of six units (light lines). The state of the ensemble units is the average of the states of its components.

value a unit takes on after probing becomes deterministic. The stable points of the system are then identical to the ones of the Hopfield and Tank model.

To prove this, it suffices to show that at each probe the ensemble Boltzmann unit takes on the state which gives rise to the lowest (Hopfield and Tank) energy. Therefore the energy must monotonically decrease. Further, if the system is not at a global (Hopfield and Tank) energy minimum, there is some unit which can be probed so as to lower the energy.

To show that the state resulting from a probe is the minimum possible, we show first that the derivative of the energy with resepect to the unit's state is zero at the resulting state, and second that the second derivative is positive over the entire range of possible states, zero to one.

Taking the derivative of equation 2 gives

$$\frac{dE}{ds_k} = -\sum_i w_{ik}s_i + \frac{1}{R_i}g^{-1}(s_k) - I_k$$

Now

$$g(u) = \frac{1}{1 + e^{-2\lambda u}}$$

so

$$g^{-1}(u) = \frac{1}{2\lambda}\ln\frac{s}{1-s}$$

Let $T = \frac{1}{2\lambda R}$. The EBM update rule is

$$s_k = \frac{1}{1 + e^{-\Delta_k/T}}$$

Therefore

$$\frac{dE}{ds_k}\bigg|_{s_k=\frac{1}{1+e^{-\Delta_k/T}}} = -\Delta_k + T \ln \left[\frac{\frac{1}{1+e^{-\Delta_k/T}}}{\frac{e^{-\Delta_k/T}}{1+e^{-\Delta_k/T}}}\right]$$

$$= -\Delta_k + T \ln e^{\Delta_k/T}$$

$$= -\Delta_k + T(\Delta_k/T)$$

$$= 0$$

and

$$\frac{d^2E}{ds_k^2} = \frac{1}{2\lambda R} \cdot \frac{1-s_k}{s_k} \cdot \left[\frac{(1-s_k)-(-s_k)}{(1-s_k)^2}\right]$$

$$= \frac{1}{2\lambda R s_k (1-s_k)}$$

$$> 0 \text{ on } 0 < s_k < 1$$

In writing a program to simulate an EBM, it would be wasteful to explicitly represent the components of each ensemble unit. Since each component has an identical energy gap, the average of their values is given by the binomial distribution $b(n,p)$ where $n$ is the ensemble size, and $p$ is $\frac{1}{1+e^{-\Delta/T}}$. There are numerical methods for sampling from this distribution in time independent of $n$ [8]. When $n$ is infinite, there is no need to bother with the distribution because the result is just $p$.

Hopfield and Tank suggest [2] that the Hopfield and Tank model is a mean field approximation to the original Hopfield model. In a mean field approximation, the average value of a variable is used to calculate its effect on other variables, rather than calculating all the individual interactions. Consider a large ensemble of Hopfield nets with two units, $A$ and $B$. To find the distribution of final states exactly, each $B$ unit must be updated based on the $A$ unit in the same network. The calculation must be repeated for every network in the ensemble. Using a mean field approximation, the average value of all the $B$ units is calculated based on the average value of all the $A$ units. This calculation is no harder than that of the state of a single Hopfield network, yet is potentially more informative since it approximates an average property of a whole ensemble of Hopfield networks. The states of Hopfield and Tank units can be viewed as representing the ensemble average of the states of Hopfield units in this way. Peterson and Anderson [9] demonstrate rigorously that the behavior is a mean field approximation.

In the EBM, it is intuitively clear that a mean field approximation is being made. The network can be thought of as a real ensemble of Boltzmann networks, except with additional connections between the networks so that each Boltzmann unit sees not only its neighbors in the same net, but also sees the average state of the neighboring units in all the nets (see figure 1).

# 6 Traveling Salesman Problem

The traveling salesman problem illustrates the use of energy-based connectionist networks, and the ease with which they may be designed. Given a list of city locations, the task is to find a tour of minimum length through all the cities and returning to the starting city. To represent a solution to an $n$ city problem in a network, it is convenient to use $n$ columns of $n$ rows of units [2]. If a unit at coordinates $(i, j)$ is on, it indicates that the $i$th city is the $j$th to be visited. A valid solution will have $n$ units on, one in every column and one in every row. The requirements can be divided into four constraints: there can be no more than one unit on in a row, no more that one unit on in a column, there must be $n$ units on, and the distances between cities must be minimized. Hopfield and Tank use the following energy function to effect these constraints:

$$E = A/2 \sum_X \sum_i \sum_{j \neq i} s_{Xi} s_{Xj} +$$
$$B/2 \sum_i \sum_X \sum_{Y \neq X} s_{Xi} s_{Yi} +$$
$$C/2 \left( \sum_X \sum_i s_{Xi} - n \right)^2 +$$
$$D/2 \sum_X \sum_{Y \neq X} \sum_i d_{XY} s_{Xi} (s_{Y,i+1} + s_{Y,i-1}) \tag{3}$$

Here units are given two subscripts to indicate their row and column, and the subscripts "wrap around" when outside the range $1 \leq i \leq n$. The first term is implemented with inhibitory links between every pair of units in a row, and is zero only if no two are on. The second term is inhibition within columns. In the third term, $n$ is the number of cities in the tour. When the system reaches a vertex of the search space, this term is zero only if exactly $n$ units are on. This constraint is implemented with inhibitory links between all $n^4$ pairs of units plus an excitatory input current to all units. In the last term $d_{XY}$ is the distance between cities $X$ and $Y$. At points in the search space representing valid tours, the summation is numerically equal to the length of the tour.

As long as the constraints ensuring that the solution is a valid tour are stronger than those minimizing distance, the global energy minimum will represent the shortest tour. However *every* valid tour will be a *local* energy minimum. Which tour is chosen will depend on the random initial starting state, and on the random probing order.

# 7 Empirical Results

The evidence that convinced me EBMs offer improved performance over Hopfield and Tank networks was the ease of tuning them for the Ted Turner problem reported

in [7]. However this evidence is entirely subjective; it is impossible to show that no set of parameters exist which would make the Hopfield and Tank model perform well. Instead we have chosen to repeat the traveling salesman problem experiments reported by Hopfield and Tank [2], using the same cities and the same values for the constants in equation 3. The tour involves 10 cities, and the shortest tour is of length 2.72. An average tour has length 4.55. Hopfield and Tank report finding a valid tour in 16 of 20 settlings, and that half of these are one of the two shortest tours.

One advantage of Hopfield and Tank nets over Boltzmann Machines is that they move continuously in the direction of the gradient. EBMs move in discrete jumps whose size is the value of the gradient along a given axis. When the system is far from equilibrium these jumps can be quite large, and the search is inefficient. Although Hopfield and Tank nets can do a whole search at high gain, Boltzmann Machines usually vary the temperature so the system can remain close to equilibrium as the low temperature equilibrium is approached. For this reason our model was more sensitive to the gain parameter than the Hopfield and Tank model, and we used temperatures much higher than $\frac{1}{2\lambda R}$.

As expected, when $n$ is infinite, an EBM produces results similar to those reported by Hopfield and Tank. 85 out of 100 settlings resulted in valid tours, and the average length was 2.73. Table 1 shows how $n$ affects the number of valid tours and the average tour length. As $n$ decreases from infinity, both the average tour length and the number of valid tours increases. (We have no explanation for the anomalously low number of valid tours for $n = 40$.) Both of these effects result from the increased sampling noise in determining the ensemble unit states for lower $n$. With more noise, the system has an easier time escaping local minima which do not represent valid tours. Yet at the same time the discriminability between the very best tours and moderately good tours decreases, because these smaller energy differences are swamped by the noise.

Rather than stop trials when the network was observed to converge, a constant number of probes, 200 per unit, was made. However we noted that convergence was generally faster for larger values of $n$. Thus for the traveling salesman problem, large $n$ give faster and better solutions, but a smaller values gives the highest reliability. Depending on the application, a value of either infinity or 50 seems best.

# 8 Conclusion

'Ensemble' Boltzmann Machines are completely upward compatible with conventional Boltzmann Machines. The above experiment can be taken to show that they perform better at the traveling salesman problem. In addition, at the limit of infinite ensemble size they perform similarly to Hopfield and Tank nets. For TSP and perhaps many other problems, the latter model seems an equally good choice. Perhaps due to the extreme regularity of the architecture, the energy space must be nicely behaved

| Ensemble Size | Percent Valid | Average Tour Length |
|---|---|---|
| 1 | 93 | 3.32 |
| 40 | 84 | 2.92 |
| 50 | 95 | 2.79 |
| 100 | 89 | 2.79 |
| 1000 | 90 | 2.80 |
| infinity | 85 | 2.73 |

Table 1: Number of valid tours out of 100 trials and average tour length, as a function of ensemble size. An ensemble size of one corresponds to a Boltzmann Machine. Infinity loosely corresponds to a Hopfield and Tank network.

in that the ravine steepness near the center of the space is a good indication of its eventual depth. In this case the ability to escape local minima is not required for good performance.

For the Ted Turner problem, which has a very irregular architecture and many more constraint types, the ability to escape local minima seems essential. Conventional Boltzmann Machines are too noisy, both for efficient search and for debugging. EBMs allow the designer the flexibility to add only as much noise as is necessary. In addition, lower noise can be used for debugging. Even though this may give poorer performance, a more deterministic search is easier for the debugger to understand, allowing the proper fix to be made.

# Acknowledgements

We appreciate receiving data and explanations from David Tank, Paul Smolensky, and Erik Sobel. This research has been supported by an ONR Graduate Fellowship, by NSF grant EET-8716324, and by the Defense Advanced Research Projects Agency (DOD), ARPA Order No. 4976 under contract F33615-87-C-1499 and monitored by the:

Avionics Laboratory
Air Force Wright Aeronautical Laboratories
Aeronautical Systems Division (AFSC)
Wright-Patterson AFB, OH 45433-6543

This research was also sponsored by the same agency under contract N00039-87-C-0251 and monitored by the Space and Naval Warfare Systems Command.

# References

[1] J. J. Hopfield, "Neural networks and physical systems with emergent collective computational abilities," *Proceedings of the National Academy of Sciences U.S.A.*, vol. 79, pp. 2554–2558, April 1982.

[2] J. Hopfield and D. Tank, "'Neural' computation of decisions in optimization problems," *Biological Cybernetics*, vol. 52, pp. 141–152, 1985.

[3] G. E. Hinton and T. J. Sejnowski, "Learning and relearning in Boltzmann Machines," in *Parallel distributed processing: Explorations in the microstructure of cognition*, Cambridge, MA: Bradford Books, 1986.

[4] S. Geman and D. Geman, "Stochastic relaxation, Gibbs distributions, and the Bayesian restoration of images," *IEEE Transactions on Pattern Analysis and Machine Intelligence*, vol. PAMI-6, pp. 721–741, 1984.

[5] J. L. Marroquin, *Probabilistic Solution of Inverse Problems*. PhD thesis, MIT, September 1985.

[6] J. Hopfield and D. Tank, "Simple 'Neural' optimization networks: an a/d converter, signal decision circuit and a linear programming circuit," *IEEE Transactions on Circuits and Systems*, vol. 33, pp. 533–541, 1986.

[7] M. Derthick, "Counterfactual reasoning with direct models," in *AAAI-87*, Morgan Kaufmann, July 1987.

[8] D. E. Knuth, *The Art of Computer Programming. Second Edition.* Vol. 2, Addison-Wesley, 1981.

[9] C. Peterson and J. R. Anderson, "A mean field theory learning algorithm for neural networks," Tech. Rep. EI-259-87, MCC, August 1987.
